# Modeling the effects of memory on human online sentence processing with particle filters

**Roger Levy**
Department of Linguistics
University of California, San Diego
rlevy@ling.ucsd.edu

**Florencia Reali    Thomas L. Griffiths**
Department of Psychology
University of California, Berkeley
{floreali,tom_griffiths}@berkeley.edu

## Abstract

Language comprehension in humans is significantly constrained by memory, yet rapid, highly incremental, and capable of utilizing a wide range of contextual information to resolve ambiguity and form expectations about future input. In contrast, most of the leading psycholinguistic models and fielded algorithms for natural language parsing are non-incremental, have run time superlinear in input length, and/or enforce structural locality constraints on probabilistic dependencies between events. We present a new limited-memory model of sentence comprehension which involves an adaptation of the particle filter, a sequential Monte Carlo method, to the problem of incremental parsing. We show that this model can reproduce classic results in online sentence comprehension, and that it naturally provides the first rational account of an outstanding problem in psycholinguistics, in which the preferred alternative in a syntactic ambiguity seems to grow more attractive over time even in the absence of strong disambiguating information.

## 1   Introduction

Nearly every sentence occurring in natural language can, given appropriate contexts, be interpreted in more than one way. The challenge of comprehending a sentence is identifying the intended intepretation from among these possibilities. More formally, each interpretation of a sentence $w$ can be associated with a structural description $T$, and to comprehend a sentence is to infer $T$ from $w$ – *parsing* the sentence to reveal its underlying structure. From a probabilistic perspective, this requires computing the posterior distribution $P(T|w)$ or some property thereof, such as the description $T$ with highest posterior probability. This probabilistic perspective has proven extremely valuable in developing both effective methods by which computers can process natural language [1, 2] and models of human language processing [3].

In real life, however, people receive nearly all linguistic input *incrementally*: sentences are spoken, and written sentences are by and large read, from beginning to end. There is considerable evidence that people also comprehend incrementally, making use of linguistic input moment by moment to resolve structural ambiguity and form expectations about future inputs [4, 5]. The incremental parsing problem can, roughly, be stated as the problem of computing the posterior distribution $P(T|w_{1...i})$ for a partial input $w_{1...i}$. To be somewhat more precise, incremental parsing involves constructing a distribution over partial structural descriptions of $w_{1...i}$ which implies the posterior $P(T|w_{1...i})$. A variety of "rational" models of online sentence processing [6, 7, 8, 9] take exactly this perspective, using the properties of $P(T|w_{1...i})$ or quantities derived from it to explain why people find some sentences more difficult to comprehend than others.

Despite their success in capturing a variety of psycholinguistic phenomena, existing rational models of online sentence processing leave open a number of questions, both theoretical and empirical. On the theoretical side, these models assume that humans are "ideal comprehenders" capable of computing $P(T|w_{1...i})$ despite its significant computational cost. This kind of idealization is common in rational models of cognition, but raises questions about how resource constraints might affect language processing. For structured probabilistic formalisms in widespread use in compu-

tational linguistics, such as probabilistic context-free grammars (PCFGs), incremental processing algorithms exist that allow the exact computation of the posterior (implicitly represented) in polynomial time [10, 11, 12], from which $k$-best structures [13] or samples from the posterior [14] can be efficiently obtained. However, these algorithms are psychologically implausible for two reasons: (1) their run time (both worst-case and practical) is superlinear in sentence length, whereas human processing time is essentially linear in sentence length; and (2) the probabilistic formalisms utilized in these algorithms impose strict locality conditions on the probabilistic dependence between events at different levels of structure, whereas humans seem to be able to make use of arbitrary features of (extra-)linguistic context in forming incremental posterior expectations [4, 5].

Theoretical questions about the mechanisms underlying online sentence processing are complemented by empirical data that are hard to explain purely in probabilistic terms. For example, one of the most compelling phenomena in psycholinguistics is that of *garden-path sentences*, such as:

(1)    The woman brought the sandwich from the kitchen tripped.

Comprehending such sentences presents a significant challenge, and many readers fail completely on their first attempt. However, the sophisticated dynamic programming algorithms typically used for incremental parsing implicitly represent all possible continuations of a sentence, and are thus able to recover the correct interpretation in a single pass. Another phenomenon that is hard to explain simply in terms of the probabilities of interpretations of a sentence is the "digging in" effect, in which the preferred alternative in a syntactic ambiguity seems to grow more attractive over time even in the absence of strong disambiguating information [15].

In this paper, we explore the hypothesis that these phenomena can be explained as the consequence of constraints on the resources available for incremental parsing. Previous work has addressed the issues of feature locality and resource constraints by adopting a pruning approach, in which hard locality constraints on probabilistic dependence are abandoned and only high-probability candidate structures are maintained after each step of incremental parsing [6, 16, 17, 18]. These approaches can be thought of as focusing on holding on to the highest posterior-probability parse as often as possible. Here, we look to the machine learning literature to explore an alternative approach focused on approximating the posterior distribution $P(T|w_{1...i})$. We use particle filters [19], a sequential Monte Carlo method commonly used for approximate probabilistic inference in an online setting, to explore how the computational resources available influence the comprehension of sentences. This approach builds on the strengths of rational models of online sentence processing, allowing us to examine how performance degrades as the resources of the ideal comprehender decrease.

The plan of the paper is as follows. Section 2 introduces the key ideas behind particle filters, while Section 3 outlines how these ideas can be applied in the context of incremental parsing. Section 4 illustrates the approach for the kind of garden-path sentence given above, and Section 5 presents an experiment with human participants testing the predictions that the resulting model makes about the digging-in effect. Section 6 concludes the paper.

## 2   Particle filters

Particle filters are a sequential Monte Carlo method typically used for probabilistic inference in contexts where the amount of data available increases over time [19]. The canonical setting in which a particle filter would be used involves a sequence of latent variables $z_1, \ldots, z_n$ and a sequence of observed variables $x_1, \ldots, x_n$, with the goal of estimating $P(z_n|x_{1...n})$. The particle filter solves this problem recursively, relying on the fact that the chain rule gives

$$P(z_n|x_{1...n}) \propto P(x_n|z_n) \sum_{z_{n-1}} P(z_n|z_{n-1}) P(z_{n-1}|x_{1...n-1}) \qquad (1)$$

where we assume $x_n$ and $z_n$ are independent of all other variables given $z_n$ and $z_{n-1}$ respectively.

Assume we know $P(z_{n-1}|x_{1...n-1})$. Then we can use this distribution to construct an importance sampler for $P(z_n|x_{1...n})$. We generate several values of $z_{n-1}$ from $P(z_{n-1}|x_{1...n-1})$. Then, we draw $z_n$ from $P(z_n|z_{n-1})$ for each instance of $z_{n-1}$, to give us a set of values from $P(z_n|x_{1...n-1})$. Finally, we assign each value of $z_n$ a weight proportional to $P(x_n|z_n)$, to give us an approximation to $P(z_n|x_{1...n})$. The particle filter is simply the recursive version of this algorithm, in which a similar approximation was used to construct to $P(z_{n-1}|x_{1...n-1})$ from $P(z_{n-2}|x_{1...n-2})$ and so forth. The algorithm thus approximates $P(z_{n-1}|x_{1...n-1})$ with a weighted set of "particles" – discrete values of $z_i$ – which are updated using $P(z_n|z_{n-1})$ and

$P(x_n|z_n)$ to provide an approximation to $P(z_n|x_{1...n})$. The particle filter thus has run-time linear in the number of observations, and provides a way to explore the influence of memory capacity (reflected in the number of particles) on probabilistic inference (cf. [20, 21]). In this paper, we focus on the conditions under which the particle filter fails as a source of information about the challenges of limited memory capacity for online sentence processing.

## 3 Incremental parsing with particle filters

In this section we develop an algorithm for top-down, incremental particle-filter parsing. We first lay out the algorithm, then consider options for representations and grammars.

### 3.1 The basic algorithm

We assume that the structural descriptions of a sentence are context-free trees, as might be produced by a PCFG. Without loss of generality, we also assume that preterminal expansions are always unary rewrites. A tree is generated incrementally in a sequence of *derivation operations* $\pi_{1...m}$, such that no word can be generated unless all the words preceding it in the sentence have already been generated. The words of the sentence can thus be considered observations, and the hidden state is a partial derivation $(D, S)$, where $D$ is an incremental tree structure and $S$ is a stack of items of the form $\langle N, \mathtt{Op} \rangle$, where $N$ is a target node in $D$ and $\mathtt{Op}$ is a derivation operation type. Later in this section, we outline three possible derivation *orders*.

The problem of inferring a distribution over partial derivations from observed words can be approximated using particle filters as outlined in Section 2. Assume a model that specifies a probability distribution $P(\pi|(D, S), w_{1...i})$ over the next derivation operation $\pi$ given the current partial derivation and words already seen. By $(D, S) \overset{\pi_{1...j}}{\Rightarrow} (D', S')$ we denote that the sequence of derivation operations $\pi_{1...j}$ takes the partial derivation $(D, S)$ to a new partial derivation $(D', S')$. Now consider a partial derivation $(D_{i|}, S_{i|})$ in which the most recent derivation operation has generated the $i^{\text{th}}$ word in the input. Through the $\overset{\pi}{\Rightarrow}$ relation, our model implies a probability distribution over new partial derivations in which the next operation would be the generation of the $i + 1^{\text{th}}$ word; call this distribution $P((D_{|i+1}, S_{|i+1})|(D_{i|}, S_{i|}))$. In the nomenclature of particle filters introduced above, partial derivations $(D_{|i}, S_{|i})$ thus correspond to latent variables $z_i$, words $w_i$ to observations $x_i$, and our importance sampler involves drawing from $P((D_{|i}, S_{|i})|(D_{i-1|}, S_{i-1|}))$ and reweighting by $P(w_i|(D_{|i}, S_{|i}))$. This differs from the standard particle filter only in that $z_i$ is not necessarily independent of $x_{1...i-1}$ given $z_{i-1}$.

### 3.2 Representations and grammars

We now describe three possible derivation orders that can be used with our approach. For each order, a derivation operation $\pi_{\mathtt{Op}}$ of a given type $\mathtt{Op}$ specifies a sequence of symbols $Y_1 \ldots Y_k$ (possibly the empty sequence $\epsilon$), and can be applied to a partial derivation: $(D, [\langle N, \mathtt{Op} \rangle] \oplus S) \overset{\pi_{\mathtt{Op}}}{\Rightarrow} (D', A \oplus S)$, with $\oplus$ being list concatenation. That is, a derivation operation involves popping the top item off the stack, choosing a derivation operation of the appropriate type, applying it to add some symbols to $D$ yielding $D'$, and pushing a list of new items $A$ back on the stack. Derivation operations differ in the relationship between $D$ and $D'$, and derivation orders differ in the contents of $A$.

Order 1: **Expansion (Exp) only.** $D'$ consists of $D$ with node $N$ expanded to have daughters $Y_1 \ldots Y_k$; and $A = [\langle Y_1, \mathtt{Exp} \rangle, \ldots, \langle Y_k, \mathtt{Exp} \rangle]$.

Order 2: **Expansion and Right-Sister (Sis).** The sequence of symbols specified by any $\pi_{\mathtt{Op}}$ is of maximum length 1. Expansion operations affect $D$ as above. For a right-sister operation $\pi_{\mathtt{Sis}}$, $D'$ consists of $D$ with $Y_1$ added as the right sister of $N$ (if $\pi_{\mathtt{Sis}}$ specifies $\epsilon$, then $D = D'$). $A = [\langle Y_1, \mathtt{Exp} \rangle, \langle Y_1, \mathtt{Sis} \rangle, \ldots, \langle Y_k, \mathtt{Exp} \rangle, \langle Y_k, \mathtt{Sis} \rangle]$.

Order 3: **Expansion, Right-Sister, and Adjunction (Adj).** The sequence of symbols specified by any $\pi_{\mathtt{Op}}$ is of maximum length 1. Expansion operations affect $D$ as above. Expansion and right-sister operations are as above. For a right-sister operation $\pi_{\mathtt{Adj}}$, $D'$ consists of $D$ with $Y_1$ spliced in at the node $N$ – that is, $Y_1$ replaces $N$ in the tree, and $N$ becomes the lone daughter of $Y_1$ (if $\pi_{\mathtt{Adj}}$ specifies $\epsilon$, then $D = D'$). $A = [\langle Y_1, \mathtt{Exp} \rangle, \langle Y_1, \mathtt{Sis} \rangle, \langle Y_1, \mathtt{Adj} \rangle, \ldots, \langle Y_k, \mathtt{Exp} \rangle, \langle Y_k, \mathtt{Sis} \rangle, \langle Y_k, \mathtt{Adj} \rangle]$.

```
        D                    S              D                 S             D                S
      ROOT              ⟨VBD, Exp⟩        ROOT          ⟨VBD, Exp⟩        ROOT         ⟨VBD, Exp⟩
       |                ⟨ADVP, Exp⟩        |            ⟨VBD, Sis⟩         |           ⟨VBD, Sis⟩
      S_1               ⟨CC, Exp⟩         S_1           ⟨VP, Sis⟩         S_2          ⟨VBD, Adj⟩
     /    \             ⟨S_3, Exp⟩         |            ⟨S_2, Sis⟩       /   \         ⟨VP, Sis⟩
   S_2    CC  S_3                         S_2           ⟨S_1, Sis⟩     NP    VP        ⟨VP, Adj⟩
  /   \                                  /   \                         |     |        ⟨S_2, Sis⟩
 NP    VP                              NP    VP                        N    VBD       ⟨S_2, Adj⟩
  |   /  \                              |     |                        |
  N VBD ADVP                            N    VBD                      Pat
  |                                     |
 Pat                                   Pat
                                                                              (c)
        (a)                                      (b)
```

Figure 1: Three possible derivation orders for the sentence "Pat walked yesterday and Sally slept". In each case, the partial derivation $(D_{|i}, S_{|i})$ is shown for $i = 2$ – up to just before the generation of the word "walked". The symbols ADVP, CC, and $S_3$ in (a) will be generated later in the derivations of (b) and (c) as right-sister operations; the symbol $S_1$ will be generated in (c) as an adjunction operation. During the incremental parsing of "walked" these partial derivations would be reweighted by $P_{\texttt{Exp}}(walked|(D_{|i}, S_{|i}))$.

In all cases, the initial state of a derivation is a root symbol targeted for expansion: $(\text{ROOT}, [\langle\text{ROOT}, \texttt{Exp}\rangle])$, and a derivation is complete when the stack is empty. Figure 1 illustrates the partial derivation state for each order just after the generation of a word in mid-sentence.

For each derivation operation type $\texttt{Op}$, it is necessary to define an underlying grammar and estimate the parameters of a distribution $P_{\texttt{Op}}(\pi|(D, S))$ over next derivation operations given the current state of the derivation. For a sentence whose tree structure is known, the sequence of derivation operations for derivation orders 1 and 2 is unambiguous and thus supervised training can be used for such a model. For derivation order 3, a known tree structure still underspecifies the order of derivation operations, so the underlying sequence of derivation operations could either be canonicalized or treated as a latent variable in training. Finally, we note that a known PCFG could be encoded in a model using any of these derivation orders; for PCFGs, the partial derivation representations used in order 3 may be thought of as marginalizing over the unary chains on the right frontier of the representations in order 2, which in turn may be thought of as marginalizing over the extra childless nonterminals in the incremental representations of order 1. In the context of the particle filter, the representations with more operation types could thus be expected to function as having larger effective sample sizes for a fixed number of particles [22]. For the experiments reported in this paper, we use derivation order 2 with a PCFG trained using unsmoothed relative-frequency estimation on the parsed Brown corpus.

This approach has several attractive features for the modeling of online human sentence comprehension. The number of particles can be considered a rough estimate of the quantity of working memory resources devoted to the sentence comprehension task; as we will show in Section 5, sentences difficult to parse can become easier when more particles are used. After each word, the incremental posterior over partial structures $T$ can be read off the particle structures and weights. Finally, the approximate *surprisal* of each word – a quantity argued to be correlated with many types of processing difficulty in sentence comprehension [8, 9, 23] – is essentially a by-product of the incremental parsing process: it is the negative log of the mean (unnormalized) weight $P(w_i|(D_{|i}, S_{|i}))$.

## 4  The garden-path sentence

To provide some intuitions about our approach, we illustrate its ability to model online disambiguation in sentence comprehension using the garden-path sentence given in Example 1. In this sentence, a local structural ambiguity is introduced at the word *brought* due to the fact that this word could be either (i) a past-tense verb, in which case it is the main verb of the sentence and *The woman* is its complete subject; or (ii) a participial verb, in which case it introduces a reduced relative clause, *The woman* is its recipient, and the subject of the main clause has not yet been completed. This ambiguity is resolved in favor of (ii) by the word *tripped*, the main verb of the sentence. It is well documented (e.g., [24]) that locally ambiguous sentences such as Example 1 are read more slowly at the disambiguating region when compared with unambiguous counterparts (c.f. *The woman who*

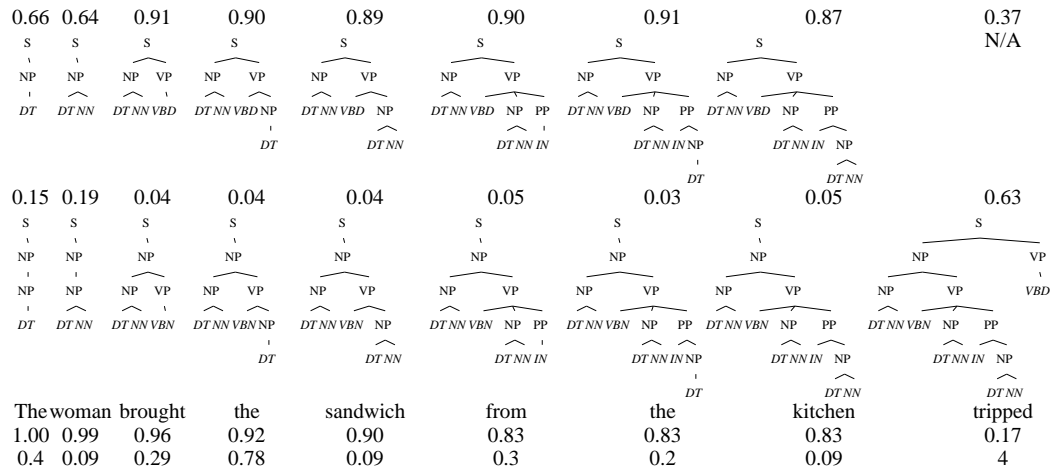

| 0.66 | 0.64 | 0.91 | 0.90 | 0.89 | 0.90 | 0.91 | 0.87 | 0.37 N/A |
| 0.15 | 0.19 | 0.04 | 0.04 | 0.04 | 0.05 | 0.03 | 0.05 | 0.63 |
| The woman | brought | the | sandwich | from | the | kitchen | tripped |
| 1.00 | 0.99 | 0.96 | 0.92 | 0.90 | 0.83 | 0.83 | 0.83 | 0.17 |
| 0.4 | 0.09 | 0.29 | 0.78 | 0.09 | 0.3 | 0.2 | 0.09 | 4 |

Figure 2: Incremental parsing of a garden-path sentence. Trees indicate the canonical structures for main-verb (above) and reduced-relative (below) interpretations. Numbers above the trees indicate the posterior probabilites of main-verb and reduced-relative interpretations, marginalizing over precise details of parse structure, as estimated by a parser using 1000 particles. Since the grammar is quite noisy, the main-verb interpretation still has some posterior probability after disambiguation at *tripped*. Numbers in the second-to-last line indicate the proportion of particle filters with 20 particles that produce a viable parse tree including the given word. The final line indicates the variance ($\times 10^{-3}$) of particle weights after parsing each word.

*was brought the sandwich from the kitchen tripped*), and in cases where the local bias strongly favors (i), many readers may fail to recover the correct reading altogether.

Figure 2 illustrates the behavior of the particle filter on the garden-path sentence in Example 1. The word *brought* shifts the posterior strongly toward the main-verb interpretation. The rest of the reduced relative clause has little effect on the posterior, but the disambiguator *tripped* shifts the posterior in favor of the correct reduced-relative interpretation. In low-memory situations, as represented by a particle filter with a small number of particles (e.g., 20), the parser is usually able to construct an interpretation for the sentence up through the word *kitchen*, but fails at the disambiguator, and when it succeeds the variance in particle weights is high.

## 5 Exploring the "digging in" phenomenon

An important feature distinguishing "rational" models of online sentence comprehension [6, 7, 8, 9] from what are sometimes called "dynamical systems" models [25, 15] is that the latter have an internal feedback mechanism: in the absence of any biasing input, the activation of the leading candidate interpretation tends to grow with the passage of time. A body of evidence exists in the psycholinguistic literature that seems to support an internal feedback mechanism: increasing the duration of a local syntactic ambiguity increases the difficulty of recovery at disambiguation to the disfavored interpretation. It has been found, for example, that 2a and 3a, in which the second NP (*the gossip. . ./the deer. . .* ) initially seems to be the object of the preceding verb, are harder to recover from than 2b and 3b [26, 27, 15].

(2)  "NP/S" ambiguous sentences

    a.  Long (A-L): Tom heard the gossip about the neighbors wasn't true.

    b.  Short (A-S): Tom heard the gossip wasn't true.

(3)  "NP/Z" ambiguous sentences

    a.  Long (A-L): While the man hunted the deer that was brown and graceful ran into the woods.

    b.  Short (A-S): While the man hunted the deer ran into the woods.

From the perspective of exact rational inference – or even for rational pruning models such as [6] – this "digging in" effect is puzzling.[1] The result finds an intuitive explanation, however, in our limited-memory particle-filter model. The probability of parse failure at the disambiguating word $w_i$ is a function of (among other things) the immediately preceding estimated posterior probability of the disfavored interpretation. If this posterior probability is low, then the resampling of particles performed after processing each word provides another point at which particles representing the disfavored interpretation could be deleted. Consequently, total parse failure at the disambiguator will become more likely the greater the length of the preceding ambiguous region.

We quantify these predictions by assuming that the more often no particle is able to integrate a given word $w_i$ in context – that is, $P(w_i|(D_{|i}, S_{|i}))$ – the more difficult, on average, people should find $w_i$ to read. In the sentences of Examples 2-3, by far the most likely position for the incremental parser to fail is at the disambiguating verb. We can also compare processing of these sentences with syntactically similar but unambiguous controls.

    (4)   "NP/S" unambiguous controls
          a.  Long (U-L): Tom heard that the gossip about the neighbors wasn't true.
          b.  Short (U-S): Tom heard that the gossip wasn't true.

    (5)   "NP/Z" unambiguous controls
          a.  Long (U-L): While the man hunted, the deer that was brown and graceful ran into the woods.
          b.  Short (U-S): While the man hunted, the deer ran into the woods.

Figure 3a shows, for each sentence of each type, the proportion of runs in which the parser successfully integrated (assigned non-zero probability to) the disambiguating verb (*was* in Example 2a and *ran* in Example 3a), among those runs in which the sentence was successfully parsed up to the preceding word. Consistent with our intuitive explanation, both the presence of local ambiguity and length of the preceding region make parse failure at the disambiguator more likely.

In the remainder of this section we test this explanation with an offline sentence acceptability study of digging-in effects. The experiment provides a way to make more detailed comparisons between the model's predictions and sentence acceptability. Consistent with the predictions of the model, ratings show differences in the magnitude of digging-in effects associated with different types of structural ambiguities. As the working-memory resources (i.e. number of particles) devoted to comprehension of the sentence increase, the probability of successful comprehension goes up, but local ambiguity and length of the second NP remain associated with greater comprehension difficulty.

## 5.1 Method

Thirty-two native English speakers from the university subject pool completed a questionnaire corresponding to the complexity-rating task. Forty experimental items were tested with four conditions per item, counterbalanced across questionnaires, plus 84 fillers, with sentence order pseudo-randomized. Twenty experimental items were NP/S sentences and twenty were NP/Z sentences. We used a $2 \times 2$ design with ambiguity and length of the ambiguous noun phrase as factors. In NP/S sentences, structural ambiguity was manipulated by the presence/absence of the complementizer *that*, while in NP/Z sentences, structural ambiguity was manipulated by the absence/presence of a comma after the first verb. Participants were asked to rate how difficult to understand sentences are on a scale from 0 to 10, 0 indicating "Very easy" and 10 "Very difficult".

## 5.2 Results and Discussion

Figure 3b shows the mean complexity rating for each type of sentences. For both NP/S and NP/Z sentences, the ambiguous long-subject (A-L) was rated the hardest to understand, and the unambiguous short-subject (U-S) condition was rated the easiest; these results are consistent with model predictions. Within sentence type, the ratings were subjected to an analysis of variance (ANOVA) with two factors: ambiguity and length. In the case of NP/S sentences there was a main effect of ambiguity, $F1(1, 31) = 12.8, p < .001, F2(1, 19) = 47.8, p < .0001$ and length, $F1(1, 31) = 4.9,$

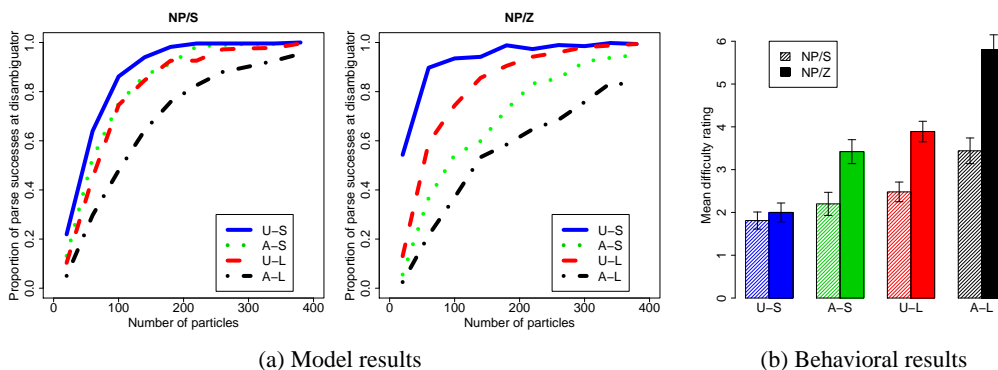

(a) Model results              (b) Behavioral results

Figure 3: Frequency of irrevocable garden path in particle-filter parser as a function of number of particles, and mean empirical difficulty rating, for NP/S and NP/Z sentences.

$p = .039$, $F2(1, 19) = 32.9$, $p < .0001$, and the interaction between factors was significant, $F1(1, 31) = 8.28$, $p = .007$, $F2(1, 19) = 5.56$, $p = .029$. In the case of NP/Z sentences there was a main effect of ambiguity, $F1(1, 31) = 63.6$, $p < .0001$, $F2(1, 19) = 150.9$, $p < .0001$ and length, $F1(1, 31) = 127.2$, $p < .0001$, $F2(1, 19) = 124.7$, $p < .0001$ and the interaction between factors was significant by subjects only, $F1(1, 31) = 4.6$, $p = .04$, $F2(1, 19) = 1.6$, $p = .2$. The experiment thus bore out most of the model's predictions, with ambiguity and length combining to make sentence processing more difficult. One reason that our model may underestimate the effect of subject length on ease of understanding, at least in the NP/Z case, is the tendency of subject NPs to be short in English, which was not captured in the grammar used by the model.

## 6 Conclusion and Future Work

In this paper we have presented a new incremental parsing algorithm based on the particle filter and shown that it provides a useful foundation for modeling the effect of memory limitations in human sentence comprehension, including a novel solution to the problem posed by "digging-in" effects [15] for rational models. In closing, we point out two issues – both involving the problem of resampling prominent in particle filter research – in which we believe future research may help deepen our understanding of language processing.

The first issue involves the question of *when to resample*. In this paper, we have take the approach of generating values of $z_{n-1}$ from which to draw $P(z_n|z_{n-1}, x_{1...n-1})$ by sampling with replacement (i.e., resampling) after every word from the multinomial over $P(z_{n-1}|x_{1...n-1})$ represented by the weighted particles. This approach has the problem that particle diversity can be lost rapidly, as it decreases monotonically with the number of observations. Another option is to resample only when the variance in particle weights exceeds a predefined threshold, sampling without replacement when this variance is low [22]. As Figure 2 shows, a word that resolves a garden-path generally creates high weight variance. Our preliminary investigations indicate that associating variance-sensitive resampling with processing difficulty leads to qualitatively similar predictions to the total parse failure approach taken in Section 5, but further investigation is required.

The other issue involves *how to resample*. Since particle diversity can never increase, when parts of the space of possible $T$ are missed by chance early on, they can never be recovered. As a consequence, applications of the particle filter in machine learning and statistics tend to supplement the basic algorithm with additional steps such as running Markov chain Monte Carlo on the particles in order to re-introduce diversity (e.g., [28]). Further work would be required, however, to specify an MCMC algorithm over trees given an input prefix. Both of these issues may help achieve a deeper understanding of the details of reanalysis in garden-path recovery [29]. For example, the initial reaction of many readers to the sentence *The horse raced past the barn fell* is to wonder what a "barn fell" is. With variance-sensitive resampling, this observation could be handled by smoothing the probabilistic grammar; with diversity-introducing MCMC, it might be handled by tree-changing operations chosen during reanalysis.

## Acknowledgments

RL would like to thank Klinton Bicknell and Gabriel Doyle for useful comments and suggestions. FR and TLG were supported by grants BCS-0631518 and BCS-070434 from the National Science Foundation.

## Footnotes

[1]For these examples, noun phrase length is a weakly misleading cue – objects tend to be longer than subjects – and that these "digging in" examples might also be analyzable as cases of exact rational inference [9]. However, the effects of length in some of the relevant experiments are quite strong. The explanation we offer here would magnify the effects of weakly misleading cues, and also extend to where cues are neutral or even favor the ultimately correct interpretation.

## References

[1] C. D. Manning and H. Schütze. *Foundations of Statistical Natural Language Processing*. MIT Press, 1999.

[2] D. Jurafsky and J. H. Martin. *Speech and Language Processing: An Introduction to Natural Language Processing, Computational Linguistics, and Speech Recognition*. Prentice-Hall, second edition, 2008.

[3] D. Jurafsky. Probabilistic modeling in psycholinguistics: Linguistic comprehension and production. In Rens Bod, Jennifer Hay, and Stefanie Jannedy, editors, *Probabilistic Linguistics*, pages 39–95. MIT Press, 2003.

[4] M. K. Tanenhaus, M. J. Spivey-Knowlton, K. Eberhard, and J. C. Sedivy. Integration of visual and linguistic information in spoken language comprehension. *Science*, 268:1632–1634, 1995.

[5] G. T. Altmann and Y. Kamide. Incremental interpretation at verbs: restricting the domain of subsequent reference. *Cognition*, 73(3):247–264, 1999.

[6] D. Jurafsky. A probabilistic model of lexical and syntactic access and disambiguation. *Cognitive Science*, 20(2):137–194, 1996.

[7] N. Chater, M. Crocker, and M. Pickering. The rational analysis of inquiry: The case for parsing. In M. Oaksford and N. Chater, editors, *Rational models of cognition*. Oxford, 1998.

[8] J. Hale. A probabilistic Earley parser as a psycholinguistic model. In *Proceedings of NAACL*, volume 2, pages 159–166, 2001.

[9] R. Levy. Expectation-based syntactic comprehension. *Cognition*, 106:1126–1177, 2008.

[10] J. Earley. An efficient context-free parsing algorithm. *Communications of the ACM*, 13(2):94–102, 1970.

[11] A. Stolcke. An efficient probabilistic context-free parsing algorithm that computes prefix probabilities. *Computational Linguistics*, 21(2):165–201, 1995.

[12] M.-J. Nederhof. The computational complexity of the correct-prefix property for TAGs. *Computational Linguistics*, 25(3):345–360, 1999.

[13] L. Huang and D. Chiang. Better $k$-best parsing. In *Proceedings of the International Workshop on Parsing Technologies*, 2005.

[14] M. Johnson, T. L. Griffiths, and S. Goldwater. Bayesian inference for PCFGs via Markov chain Monte Carlo. In *Proceedings of Human Language Technologies 2007: The Conference of the North American Chapter of the Association for Computational Linguistics*, 2007.

[15] W. Tabor and S. Hutchins. Evidence for self-organized sentence processing: Digging in effects. *Journal of Experimental Psychology: Learning, Memory, and Cognition,*, 30(2):431–450, 2004.

[16] B. Roark. Probabilistic top-down parsing and language modeling. *Computational Linguistics*, 27(2):249–276, 2001.

[17] M. Collins and B. Roark. Incremental parsing with the perceptron algorithm. In *Proceedings of the ACL*, 2004.

[18] J. Henderson. Lookahead in deterministic left-corner parsing. In *Proceedings of the Workshop on Incremental Parsing: Bringing Engineering and Cognition Together*, 2004.

[19] A. Doucet, N. de Freitas, and N. Gordon, editors. *Sequential Monte Carlo Methods in Practice*. Springer, 2001.

[20] A. N. Sanborn, T. L. Griffiths, and D. J. Navarro. A more rational model of categorization. In *Proceedings of the 28th Annual Conference of the Cognitive Science Society*, Mahwah, NJ, 2006. Erlbaum.

[21] N. Daw and A. Courville. The pigeon as particle filter. In *Advances in Neural Information Processing Systems 20*, Cambridge, MA, 2008. MIT Press.

[22] A. Doucet, N. de Freitas, K. Murphy, and S. Russell. Rao-Blackwellised particle filtering for dynamic Bayesian networks. In *Advances in Neural Information Processing Systems*, 2000.

[23] N. Smith and R. Levy. Optimal processing times in reading: a formal model and empirical investigation. In *Proceedings of the 30th Annual Meeting of the Cognitive Science Society*, 2008.

[24] M. C. MacDonald. Probabilistic constraints and syntactic ambiguity resolution. *Language and Cognitive Processes*, 9(2):157–201, 1994.

[25] M. J. Spivey and M. K. Tanenhaus. Syntactic ambiguity resolution in discourse: Modeling the effects of referential content and lexical frequency. *Journal of Experimental Psychology: Learning, Memory, and Cognition*, 24(6):1521–1543, 1998.

[26] L. Frazier and K. Rayner. Making and correcting errors during sentence comprehension: Eye movements in the analysis of structurally ambiguous sentences. *Cognitive Psychology*, 14:178–210, 1982.

[27] F. Ferreira and J. M. Henderson. Recovery from misanalyses of garden-path sentences. *Journal of Memory and Language*, 31:725–745, 1991.

[28] N. Chopin. A sequential particle filter method for static models. *Biometrika*, 89:539–552, 2002.

[29] P. Sturt, M. J. Pickering, and M. W. Crocker. Structural change and reanalysis difficulty in language comprehension. *Journal of Memory and Language*, 40:143–150, 1999.
